# Learning Time-varying Concepts

**Anthony Kuh**
Dept. of Electrical Eng.
U. of Hawaii at Manoa
Honolulu, HI 96822
kuh@wiliki.eng.hawaii.edu

**Thomas Petsche**
Siemens Corp. Research
755 College Road East
Princeton, NJ 08540
petsche@learning.siemens.com

**Ronald L. Rivest**
Lab. for Computer Sci.
MIT
Cambridge, MA 02139
rivest@theory.lcs.mit.edu

## Abstract

This work extends computational learning theory to situations in which concepts vary over time, e.g., system identification of a time-varying plant. We have extended formal definitions of concepts and learning to provide a framework in which an algorithm can track a concept as it evolves over time. Given this framework and focusing on memory-based algorithms, we have derived some PAC-style sample complexity results that determine, for example, when tracking is feasible. We have also used a similar framework and focused on incremental tracking algorithms for which we have derived some bounds on the mistake or error rates for some specific concept classes.

## 1 INTRODUCTION

The goal of our ongoing research is to extend computational learning theory to include concepts that can change or evolve over time. For example, face recognition is complicated by the fact that a persons face changes slowly with age and more quickly with changes in make up, hairstyle, or facial hair. Speech recognition is complicated by the fact that a speakers voice may change over time due to fatigue, illness, stress, or background noise (Galletti and Abbott, 1989).

Time varying systems often appear in adaptive control or signal processing applications. For example, adaptive equalizers adjust the receiver and transmitter to compensate for changes in the noise on a transmission channel (Lucky et al., 1968). The kinematics of a robot arm can change when it picks up a heavy load or when the motors and drive train responses change due to wear. The output of a sensor may drift over time as the components age or as the temperature changes.

Computational learning theory as introduced by Valiant (1984) can make some useful statements about whether a given class of concepts can be learned and provide probabilistic bounds on the number of examples needed to learn a concept. Haussler, et al. (1987), and Littlestone (1989) have also shown that it is possible to bound the number of mistakes that a learner will make. However, while these analyses allow the concept to be chosen arbitrarily, that concept must remain fixed for all time. Littlestone and Warmuth (1989) considered concepts that may drift, but in the context of a different accuracy measure than we use. Our research seeks explore further modifications to existing theory to allow the analysis of performance when learning time-varying concept.

In the following, we describe two approaches we are exploring. Section 3 describes an extension of the PAC-model to include time-varying concepts and shows how this new model applies to algorithms that base their hypotheses on a set of stored examples. Section 4 described how we can bound the mistake rate of an algorithm that updates its estimate based on the most recent example. In Section 2 we define some notation and terminology that is used in the remainder of the based.

## 2   NOTATION & TERMINOLOGY

For a dichotomy that labels each instance as a positive or negative example of a concept, we can formally describe the model as follows. Each instance $x_i$ is drawn randomly, according to an arbitrary fixed probability distribution, from an instance space $X$. The concept $c$ to be learned is drawn randomly, according to an arbitrary fixed probability distribution, from a concept class $C$. Associated with each instance is a label $a_i = c(x_i)$ such that $a_i = 1$ if $x_i$ is a positive example and $a_i = 0$ otherwise. The learner is presented with a sequence of examples (each example is a pair $\langle x_i, a_i \rangle$) chosen randomly from $X$. The learner must form an estimate, $\hat{c}$, of $c$ based on these examples.

In the time-varying case, we assume that there is an adversary who can change $c$ over time, so we change notation slightly. The instance $x_t$ is presented at time $t$. The concept $c_t$ is *active* at time $t$ if the adversary is using $c_t$ to label instances at that time. The sequence of $t$ active concepts, $\mathbf{c}_t = \{c_1, \ldots, c_t\}$ is called a *concept sequence of length $t$*. The algorithm's task is to form an estimate $\hat{\mathbf{c}}_t$ of the actual concept sequence $\mathbf{c}_t$, i.e., at each time $t$, the tracker must use the sequence of randomly chosen examples to form an estimate $\hat{c}_t$ of $c_t$. A set of length $t$ concept sequences is denoted by $\mathcal{C}(t)$ and we call a set of infinite length concept sequences a *concept sequence space* and denote it by $\mathcal{C}$.

Since the adversary, if allowed to make arbitrary changes, can easily make the tracker's task impossible, it is usually restricted such that only small or infrequent changes are allowed. In other words, each $\mathcal{C}(t)$ is a small subset of $C^t$.

We consider two different types of different types of "tracking" (learning) algorithms, memory-based (or batch) and incremental (or on-line). We analyze the sample complexity of batch algorithms and the mistake (or error) rate of incremental algorithms.

In the usual case where concepts are time-invariant, batch learning algorithms operate in two distinct phases. During the first phase, the algorithm collects a set of training examples. Given this set, it then computes a hypothesis. In the second phase, this hypothesis is used to classify all future instances. The hypothesis is never again updated. In Section 3 we consider memory-based algoritms derived from batch algorithms.

When concepts are time-invariant, an on-line learning algorithm is one which constantly modifies its hypothesis. On each iteration, the learner (1) receives an instance; (2) predicts a label based on the current hypothesis; (3) receives the correct label; and (4) uses the correct label to update the hypothesis. In Section 4, we consider incremental algorithms based on on-line algorithms.

When studying learnability, it is helpful to define the Vapnik-Chervonenkis (VC) dimension (Vapnik and Chervonenkis, 1971) of a concept class: VCdim($C$) is the cardinality of the largest set such that every possible labeling scheme is achieved by some concept in $C$. Blumer et al. (1989) showed that a concept class is learnable if and only if the VC-dimension is finite and derived an upper bound (that depends on the VC dimension) for the number of examples need to PAC-learn a learnable concept class.

## 3  MEMORY-BASED TRACKING

In this section, we will consider memory-based trackers which base their current hypothesis on a stored set of examples. We build on the definition of PAC-learning to define what it means to PAC-track a concept sequence. Our main result here is a lower bound on the maximum rate of change that can be PAC-tracked by a memory-based learner.

A *memory-based tracker* consists of (a) a function $w(\epsilon, \delta)$; and (b) an algorithm $\mathcal{L}$ that produces the current hypothesis, $\hat{c}_t$ using the most recent $w(\epsilon, \delta)$ examples. The memory-based tracker thus maintains a sliding window on the examples that includes the most recent $w(\epsilon, \delta)$ examples. We do not require that $\mathcal{L}$ run in polynomial time.

Following the work of Valiant (1984) we say that an algorithm $\mathcal{A}$ *PAC-tracks a concept sequence space* $C' \subseteq C$ if, for any $\mathbf{c} \in C'$, any distribution $D$ on $X$, any $\epsilon, \delta > 0$, and access to examples randomly selected from $X$ according to $D$ and labeled at time $t$ by concept $c_t$; for all $t$ sufficiently large, with $t'$ chosen uniformly at random between 1 and $t$, it is true that

$$\Pr(d(c_{t'}, \hat{c}_{t'}) \leq \epsilon) \geq 1 - \delta.$$

The probability includes any randomization algorithm $\mathcal{A}$ may use as well as the random selection of $t'$ and the random selection of examples according to the distribution $D$, and where $d(c, c') = D(x : c(x) \neq c'(x))$ is the probability that $c$ and $c'$ disagree on a randomly chosen example.

Learnability results often focus on learners that see only positive examples. For many concept classes this is sufficient, but for others negative examples are also necessary. Natarajan (1987) showed that a concept class that is PAC-learnable can be learned using only positive examples if the class is closed under intersection.

With this in mind, let's focus on a memory-based tracker that modifies its estimate using only positive examples. Since PAC-tracking requires that $\mathcal{A}$ be able to PAC-learn individual concepts, it must be true that $\mathcal{A}$ can PAC-track a sequence of concepts only if the concept class is closed under intersection. However, this is not sufficient.

**Observation 1.** *Assume $C$ is closed under intersection. If positive examples are drawn from $c_1 \in C$ prior to time $t_0$, and from $c_2 \in C$, $c_1 \subseteq c_2$, after time $t_0$, then there exists an estimate of $c_2$ that is consistent with all examples drawn from $c_1$.*

The proof of this is straightforward once we realize that if $c_1 \subseteq c_2$, then all positive

examples drawn prior to time $t_0$ from $c_1$ are consistent with $c_2$. The problem is therefore equivalent to first choosing a set of examples from a subset of $c_2$ and then choosing more examples from all of $c_2$ — it skews that probability distribution, but any estimate of $c_2$ will include all examples drawn from $c_1$.

Consider the set of closed intervals on $[0, 1]$, $C = \{[a, b] \mid 0 \le a, b \le 1\}$. Assume that, for some $d \ge b$, $c_t = c_1 = [a, b]$ for all $t \le t_0$ and $c_t = c_2 = [a, d]$ for all $t > t_0$. All the examples drawn prior to $t_0$, $\{x_t : t < t_0\}$, are consistent with $c_2$ and it would be nice to use these examples to help estimate $c_2$. How much can these examples help?

**Theorem 1.** *Assume $C$ is closed under intersection and $\mathrm{VCdim}(C)$ is finite; $C_2 \subseteq C$; and $A$ has PAC learned $c_1 \in C$ at time $t_0$. Then, for some $d$ such that $\mathrm{VCdim}(C_2) \le d \le \mathrm{VCdim}(C)$, the maximum number of examples drawn after time $t_0$ required so that $A$ can PAC learn $c_2 \in C$ is upper bounded by $m(\epsilon, \delta) = \max\left(\frac{4}{\epsilon} \log \frac{2}{\delta}, \frac{8d}{\epsilon} \log \frac{13}{\epsilon}\right)$*

In other words, if there is no prior information about $c_2$, then the number of examples required depends on $\mathrm{VCdim}(C)$. However, the examples drawn from $c_1$ can be used to shrink the concept space towards $C_2$. For example, when $c_1 = [a, b]$ and $c_2 = [a, c]$, in the limit where $c_1' = c_1$, the problem of learning $c_2$ reduces to learning a one-sided interval which has VC-dimension 1 versus 2 for the two-sided interval. Since it is unlikely that $c_1' = c_1$, it will usually be the case that $d > \mathrm{VCdim}(C_2)$.

In order to PAC-track $c$, most of the time $A$ must have $m(\epsilon, \delta)$ examples consistent with the current concept. This implies that $w(\epsilon, \delta)$ must be at least $m(\epsilon, \delta)$. Further, since the concepts are changing, the consistent examples will be the most recent. Using a sliding window of size $m(\epsilon, \delta)$, the tracker will have an estimate that is based on examples that are consistent with the active concept after collecting no more than $m(\epsilon, \delta)$ examples after a change.

In much of our analysis of memory-based trackers, we have focused on a concept sequence space $C_\lambda$ which is the set of all concept sequences such that, on average, each concept is active for at least $1/\lambda$ time steps before a change occurs. That is, if $N(c, t)$ is the number of changes in the first $t$ time steps of $c$, $C_\lambda = \{c : \limsup_{t \to \infty} N(c, t)/t \le \lambda\}$. The question then is, for what values of $\lambda$ does there exist a PAC-tracker?

**Theorem 2.** *Let $\mathcal{L}$ be a memory-based tracker with $w(\epsilon, \delta) = m(\epsilon, \delta/2)$ which draws instances labeled according to some concept sequence $c \in C_\lambda$ with each $c_t \in C$ and $\mathrm{VCdim}(C) < \infty$. For any $\epsilon > 0$ and $\delta > 0$, $A$ can UPAC track $C$ if $\lambda < \frac{\delta}{2} m(\epsilon, \delta/2)$.*

This theorem provides a lower bound on the maximum rate of change that can be tracked by a batch tracker. Theorem 1 implies that a memory-based tracker can use examples from a previous concept to help estimate the active concept. The proof of theorem 2 assumes that some of the most recent $m(\epsilon, \delta)$ examples are not consistent with $c_t$ until $m(\epsilon, \delta)$ examples from the active concept have been gathered. An algorithm that removes inconsistent examples more intelligently, e.g., by using conflicts between examples or information about allowable changes, will be able to track concept sequence spaces that change more rapidly.

## 4  INCREMENTAL TRACKING

Incremental tracking is similar to the on-line learning case, but now we assume that there is an adversary who can change the concept such that $c_{t+1} \neq c_t$. At each iteration:

1. the adversary chooses the active concept $c_t$;

2. the tracker is given an unlabeled instance, $x_t$;

3. the tracker predicts a label using the current hypothesis: $\hat{a}_t = \hat{c}_{t-1}(x_t)$;

4. the tracker is given the correct label $a_t$;

5. the tracker forms a new hypothesis: $\hat{c}_t = \mathcal{L}(\hat{c}_{t-1}, \langle x_t, a_t \rangle)$.

We have defined a number of different types of trackers and adversaries: A *prudent* tracker predicts that $a_t = 1$ if and only if $\hat{c}_t(x_t) = 1$. A *conservative tracker* changes its hypothesis only if $a_t \neq \hat{a}_t$. A *benign adversary* changes the concept in a way that is independent of the tracker's hypothesis while a *malicious adversary* uses information about the tracker and its hypothesis to choose a $c_{t+1}$ to cause an increase in the error rate. The *most malicious adversary* chooses $c_{t+1}$ to cause the largest possible increase in error rate on average.

We distinguish between the error of the hypothesis formed in step 5 above and a mistake made in step 3 above. The *instantaneous error rate* of an hypothesis is $e_t = d(c_t, \hat{c}_t)$. It is the probability that another randomly chosen instance labeled according to $c_t$ will be misclassified by the updated hypothesis. A *mistake* is a mislabeled instance, and we define a mistake indicator function $M_t = 1$ if $c_t(x_t) \neq \hat{c}_{t-1}(x_t)$.

We define the *average error rate* $\varepsilon_t = \frac{1}{t} \sum_{i=1}^{t} e_t$ and the *asymptotic error rate* is $\varepsilon = \lim\inf_{t \to \infty} \varepsilon_t$. The *average mistake rate* is the average value of the mistake indicator function, $\mu_t = \frac{1}{t} \sum_{i=1}^{t} M_t$, and the *asymptotic mistake rate* is $\mu = \lim\inf_{t \to \infty} \mu_t$.

We are modeling the incremental tracking problems as a Markov process. Each state of the Markov process is labeled by a triple $\langle c, \hat{c}, \alpha \rangle$, and corresponds to an iteration in which $c$ is the active concept, $\hat{c}$ is the active hypothesis, and $\alpha$ is the set of changes the adversary is allowed to make given $c$. We are still in the process of analyzing a general model, so the following presents one of the special cases we have examined.

Let $X$ be the set of all points on the unit circle. We use polar coordinates so that since the radius is fixed we can label each point by an angle $\theta$, thus $X = [0, 2\pi)$. Note that $X$ is periodic. The concept class $C$ is the set of all arcs of fixed length $\pi$ radians, i.e., all semicircles that lie on the unit circle. Each $c \in C$ can be written as $c = [\pi(2\theta - 1) \bmod 2\pi, 2\pi\theta)$, where $\theta \in [0, 1)$. We assume that the instances are chosen uniformly from the circle.

The adversary may change the concept by rotating it around the circle, however, the maximum rotation is bounded such that, given $c_t$, $c_{t+1}$ must satisfy $d(c_{t+1}, c_t) \leq \gamma$. For the uniform case, this is equivalent to restricting $\theta_{t+1} = \theta_t \pm \beta \bmod 1$, where $0 \leq \beta \leq \gamma/2$.

The tracker is required to be conservative, but since we are satisfied to lower bound the error rate, we assume that every time the tracker makes a mistake, it is told the correct concept. Thus, $\hat{c}_t = \hat{c}_{t-1}$ if no mistake is made, but $\hat{c}_t = c_t$ wherever a mistake is made.

The worst case or most malicious adversary for a conservative tracker always tries to maximize the tracker's error rate. Therefore, whenever the tracker deduces $c_t$ (i.e. whenever the tracker makes a mistake), the adversary picks a direction by flipping a fair coin. The adversary then rotates the concept in that direction as far as possible on each iteration. Then we can define a random direction function $S_t$ and write

$$S_t = \begin{cases} +1, & \text{w.p. } 1/2 \text{ if } \hat{c}_{t-1} = c_{t-1}; \\ -1, & \text{w.p. } 1/2 \text{ if } \hat{c}_{t-1} = c_{t-1}; \\ S_{t-1}, & \text{if } \hat{c}_{t-1} \neq c_{t-1}. \end{cases}$$

Then the adversary chooses the new concept to be $\theta_t = \theta_{t-1} + S_t \gamma/2$.

Since the adversary always rotates the concept by $\gamma/2$, there are $2/\gamma$ distinct concepts that can occur. However, when $\theta(t+1/\gamma) = \theta(t)+1/2 \mod 1$, the semicircles do not overlap and therefore, after at most $1/\gamma$ changes, a mistake will be made with probability one. Because at most $1/\gamma$ consecutive changes can be made before the mistake rate returns to zero, because the probability of a mistake depends only on $\theta_t - \hat{\theta}_t$, and because of inherent symmetries, this system can be modeled by a Markov chain with $k = 1/\gamma$ states. Each state $s_i$ corresponds to the case $|\theta_t - \hat{\theta}_t| = i\gamma \mod 1$. The probability of a transition from state $s_i$ to state $s_{i+1}$ is $P(s_{i+1}|s_i) = 1 - (i+1)\gamma$. The probability of a transition from state $s_i$ to state $s_0$ is $P(s_0|s_i) = (i+1)\gamma$. All other transition probabilities are zero. This Markov chain is homogeneous, irreducible, aperiodic, and finite so it has an invariant distribution. By solving the balance equations, for $\gamma$ sufficiently small, we find that

$$P(s_l) = \frac{\prod_{i=0}^{l}(1 - i\gamma)}{\sum_{j=0}^{\frac{1}{\gamma}-1} \prod_{i=0}^{j}(1 - i\gamma)} \approx \sqrt{2\gamma/\pi} \int_{l}^{l+1} e^{-\frac{\gamma}{2}x^2} \, dx \tag{1}$$

Since we assume that $\gamma$ is small, the probability that no mistake will occur for each of $k - 1$ consecutive time steps after a mistake, $P(s_{k-1})$, is very small and we can say that the probability of a mistake is approximately $P(s_0)$. Therefore, from equation 1, for small $\gamma$, it follows that $\mu_{\text{malicious}} \approx \sqrt{2\gamma/\pi}$.

If we drop the assumption that the adversary is malicious, and instead assume the the adversary chooses the direction randomly at each iteration, then a similar sort of analysis yields that $\mu_{\text{benign}} = O(\gamma^{2/3})$.

Since the foregoing analysis assumes a conservative tracker that chooses the best hypothesis every time it makes a mistake, it implies that for this concept sequence space and *any* conservative tracker, the mistake rate is $O(\gamma^{1/2})$ against a malicious adversary and $O(\gamma^{2/3b})$ against a benign adversary. For either adversary, it can be shown that $\varepsilon = \mu - \gamma$.

## 5   CONCLUSIONS AND FURTHER RESEARCH

We can draw a number of interesting conclusions form the work we have done so far. First, tracking sequences of concepts is possible when the individual concepts are learnable and change occurs "slowly" enough. Theorem 2 gives a weak upper bound on the rate of concept changes that is sufficient to insure that tracking is possible.

Theorem 1 implies that there can be some trade-off between the size (VC-dimension) of the changes and the rate of change. Thus, if the size of the changes is restricted, Theorems 1 and 2 together imply that the maximum rate of change can be faster than for the general case. It is significant that a simple tracker that maintains a sliding window on the most recent set of examples can PAC-track the new concept after a change as quickly as a static learner can if it starts from scratch. This suggests it may be possible to subsume detection so that it is implicit in the operation of the tracker. One obviously open problem is to determine $d$ in Theorem 1, i.e., what is the appropriate dimension to apply to the concept changes?

The analysis of the mistake and error rates presented in Section 4 is for a special case with VC-dimension 1, but even so, it is interesting that the mistake and error rates are significantly worse than the rate of change. Preliminary analysis of other concept classes suggests that this continues to be true for higher VC-dimensions. We are continuing work to extend this analysis to other concept classes, including classes with higher VC-dimension; non-conservative learners; and other restrictions on concept changes.

## Acknowledgments

Anthony Kuh gratefully acknowledges the support of the National Science Foundation through grant EET-8857711 and Siemens Corporate Research. Ronald L. Rivest gratefully acknowledges support from NSF grant CCR-8914428, ARO grant N00014-89-J-1988, and a grant from the Siemens Corporation.

## References

Blumer, A., Ehrenfeucht, A., Haussler, D., and Warmuth, M. (1989). Learnability and the Vapnik-Chervonenkis dimension. *Journal of the Association for Computing Machinery*, 36(4):929–965.

Galletti, I. and Abbott, M. (1989). Development of an advanced airborne speech recognizer for direct voice input. *Speech Technology*, pages 60–63.

Haussler, D., Littlestone, N., and Warmuth, M. K. (1987). Expected mistake bounds for on-line learning algorithms. (Unpublished).

Littlestone, N. (1989). Mistake bounds and logarithmic linear-threshold learning algorithms. Technical Report UCSC-CRL-89-11, Univ. of California at Santa Cruz.

Littlestone, N. and Warmuth, M. K. (1989). The weighted majority algorithm. In *Proceedings of IEEE FOCS Conference*, pages 256–261. IEEE. (Extended abstract only.).

Lucky, R. W., Salz, J., and Weldon, E. J. (1968). *Principles of Data Communications*. McGraw-Hill, New York.

Natarajan, B. K. (1987). On learning boolean functions. In *Proceedings of the Nineteenth Annual ACM Symposium on Theory of Computing*, pages 296–304.

Valiant, L. (1984). A theory of the learnable. *Communications of the ACM*, 27:1134–1142.

Vapnik, V. N. and Chervonenkis, A. Y. (1971). On the uniform convergence of relative frequencies of events to their probabilities. *Theory of Probability and its Applications*, 16:264–280.